# Information Bottleneck Optimization and Independent Component Extraction with Spiking Neurons

**Stefan Klampfl, Robert Legenstein, Wolfgang Maass**
Institute for Theoretical Computer Science
Graz University of Technology
A-8010 Graz, Austria
{klampfl,legi,maass}@igi.tugraz.at

## Abstract

The extraction of statistically independent components from high-dimensional multi-sensory input streams is assumed to be an essential component of sensory processing in the brain. Such independent component analysis (or blind source separation) could provide a less redundant representation of information about the external world. Another powerful processing strategy is to extract preferentially those components from high-dimensional input streams that are related to other information sources, such as internal predictions or proprioceptive feedback. This strategy allows the optimization of internal representation according to the information bottleneck method. However, concrete learning rules that implement these general unsupervised learning principles for spiking neurons are still missing. We show how both information bottleneck optimization and the extraction of independent components can in principle be implemented with stochastically spiking neurons with refractoriness. The new learning rule that achieves this is derived from abstract information optimization principles.

## 1 Introduction

The Information Bottleneck (IB) approach and independent component analysis (ICA) have both attracted substantial interest as general principles for unsupervised learning [1, 2]. A hope has been, that they might also help us to understand strategies for unsupervised learning in biological systems. However it has turned out to be quite difficult to establish links between known learning algorithms that have been derived from these general principles, and learning rules that could possibly be implemented by synaptic plasticity of a spiking neuron. Fortunately, in a simpler context a direct link between an abstract information theoretic optimization goal and a rule for synaptic plasticity has recently been established [3]. The resulting rule for the change of synaptic weights in [3] maximizes the mutual information between pre- and postsynaptic spike trains, under the constraint that the postsynaptic firing rate stays close to some target firing rate. We show in this article, that this approach can be extended to situations where simultaneously the mutual information between the postsynaptic spike train of the neuron and other signals (such as for example the spike trains of other neurons) has to be minimized (Figure 1). This opens the door to the exploration of learning rules for information bottleneck analysis and independent component extraction with spiking neurons that would be optimal from a theoretical perspective.

We review in section 2 the neuron model and learning rule from [3]. We show in section 3 how this learning rule can be extended so that it not only maximizes mutual information with some given spike trains and keeps the output firing rate within a desired range, but simultaneously minimizes mutual information with other spike trains, or other time-varying signals. Applications to infor-

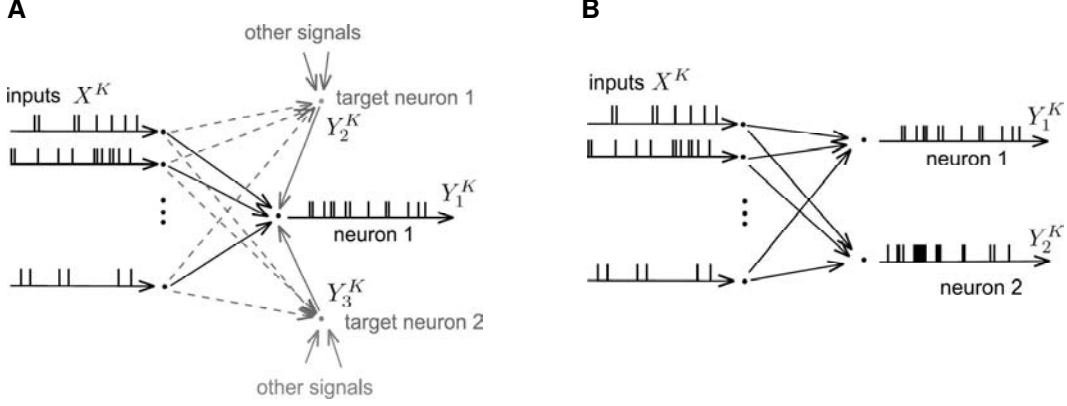

Figure 1: Different learning situations analyzed in this article. **A** In an information bottleneck task the learning neuron (neuron 1) wants to maximize the mutual information between its output $Y_1^K$ and the activity of one or several target neurons $Y_2^K, Y_3^K, \dots$ (which can be functions of the inputs $X^K$ and/or other external signals), while at the same time keeping the mutual information between the inputs $X^K$ and the output $Y_1^K$ as low as possible (and its firing rate within a desired range). Thus the neuron should learn to extract from its high-dimensional input those aspects that are related to these target signals. This setup is discussed in sections 3 and 4. **B** Two neurons receiving the same inputs $X^K$ from a common set of presynaptic neurons both learn to maximize information transmission, and simultaneously to keep their outputs $Y_1^K$ and $Y_2^K$ statistically independent. Such extraction of independent components from the input is described in section 5.

mation bottleneck tasks are discussed in section 4. In section 5 we show that a modification of this learning rule allows a spiking neuron to extract information from its input spike trains that is independent from the component extracted by another neuron.

## 2 Neuron model and a basic learning rule

We use the model from [3], which is a stochastically spiking neuron model with refractoriness, where the probability of firing in each time step depends on the current membrane potential and the time since the last output spike. It is convenient to formulate the model in discrete time with step size $\Delta t$. The total membrane potential of a neuron $i$ in time step $t^k = k\Delta t$ is given by

$$u_i(t^k) = u_r + \sum_{j=1}^{N} \sum_{n=1}^{k} w_{ij}\epsilon(t^k - t^n)x_j^n, \tag{1}$$

where $u_r = -70\text{mV}$ is the resting potential and $w_{ij}$ is the weight of synapse $j$ ($j = 1, \dots, N$). An input spike train at synapse $j$ up to the $k$-th time step is described by a sequence $X_j^k = (x_j^1, x_j^2, \dots, x_j^k)$ of zeros (no spike) and ones (spike); each presynaptic spike at time $t^n$ ($x_j^n = 1$) evokes a postsynaptic potential (PSP) with exponentially decaying time course $\epsilon(t - t^n)$ with time constant $\tau_m = 10\text{ms}$. The probability $\rho_i^k$ of firing of neuron $i$ in each time step $t^k$ is given by

$$\rho_i^k = 1 - \exp[-g(u_i(t^k))R_i(t^k)\Delta t] \approx g(u_i(t^k))R_i(t^k)\Delta t, \tag{2}$$

where $g(u) = r_0 \log\{1 + \exp[(u - u_0)/\Delta u]\}$ is a smooth increasing function of the membrane potential $u$ ($u_0 = -65\text{mV}$, $\Delta u = 2\text{mV}$, $r_0 = 11\text{Hz}$). The approximation is valid for sufficiently small $\Delta t$ ($\rho_i^k \ll 1$). The refractory variable $R_i(t) = \frac{(t-\hat{t}_i-\tau_{abs})^2}{\tau_{refr}^2+(t-\hat{t}_i-\tau_{abs})^2}\Theta(t - \hat{t}_i - \tau_{abs})$ assumes values in $[0, 1]$ and depends on the last firing time $\hat{t}_i$ of neuron $i$ (absolute refractory period $\tau_{abs} = 3\text{ms}$, relative refractory time $\tau_{refr} = 10\text{ms}$). The Heaviside step function $\Theta$ takes a value of 1 for non-negative arguments and 0 otherwise.

This model from [3] is a special case of the spike-response model, and with a refractory variable $R(t)$ that depends only on the time since the last postsynaptic event it has renewal properties [4].

The output of neuron $i$ at the $k$-th time step is denoted by a variable $y_i^k$ that assumes the value 1 if a postsynaptic spike occurred and 0 otherwise. A specific spike train up to the $k$-th time step is written as $Y_i^k = (y_i^1, y_i^2, \ldots, y_i^k)$.

The information transmission between an ensemble of input spike trains $\mathbf{X}^K$ and the output spike train $\mathbf{Y}_i^K$ can be quantified by the mutual information[1] [5]

$$I(\mathbf{X}^K; \mathbf{Y}_i^K) = \sum_{X^K, Y_i^K} P(X^K, Y_i^K) \log \frac{P(Y_i^K | X^K)}{P(Y_i^K)}. \tag{3}$$

The idea in [3] was to maximize the quantity $I(\mathbf{X}^K; \mathbf{Y}_i^K) - \gamma D_{KL}(P(Y_i^K) || \tilde{P}(Y_i^K))$, where $D_{KL}(P(Y_i^K) || \tilde{P}(Y_i^K)) = \sum_{Y_i^K} P(Y_i^K) \log(P(Y_i^K)/\tilde{P}(Y_i^K))$ denotes the Kullback-Leibler divergence [5], imposing the additional constraint that the firing statistics $P(Y_i)$ of the neuron should stay as close as possible to a target distribution $\tilde{P}(Y_i)$. This distribution was chosen to be that of a constant target firing rate $\tilde{g}$ accounting for homeostatic processes. An online learning-rule performing gradient ascent on this quantity was derived for the weights $w_{ij}$ of neuron $i$, with $\Delta w_{ij}^k$ denoting the weight change during the $k$-th time step:

$$\frac{\Delta w_{ij}^k}{\Delta t} = \alpha C_{ij}^k B_i^k(\gamma), \tag{4}$$

which consists of the "correlation term" $C_{ij}^k$ and the "postsynaptic term" $B_i^k$ [3]. The term $C_{ij}^k$ measures coincidences between postsynaptic spikes at neuron $i$ and PSPs generated by presynaptic action potentials arriving at synapse $j$,

$$C_{1j}^k = C_{1j}^{k-1} \left(1 - \frac{\Delta t}{\tau_C}\right) + \sum_{n=1}^{k} \epsilon(t^k - t^n) x_j^n \frac{g'(u_1(t^k))}{g(u_1(t^k))} \left[y_1^k - \rho_1^k\right], \tag{5}$$

in an exponential time window with time constant $\tau_C = 1$s and $g'(u_i(t^k))$ denoting the derivative of $g$ with respect to $u$. The term

$$B_1^k(\gamma) = \frac{y_1^k}{\Delta t} \log \left[\frac{g(u_1(t^k))}{\bar{g}_1(t^k)} \left(\frac{\tilde{g}}{\bar{g}_1(t^k)}\right)^{\gamma}\right]$$
$$- (1 - y_1^k) R_1(t^k) \left[g(u_1(t^k)) - (1 + \gamma)\bar{g}_1(t^k) + \gamma \tilde{g}\right], \tag{6}$$

compares the current firing rate $g(u_i(t^k))$ with its average firing rate[2] $\bar{g}_i(t^k)$, and simultaneously the running average $\bar{g}_i(t^k)$ with the constant target rate $\tilde{g}$. The argument indicates that this term also depends on the optimization parameter $\gamma$.

## 3  Learning rule for multi-neuron interactions

We extend the learning rule presented in the previous section to a more complex scenario, where the mutual information between the output spike train $Y_1^K$ of the learning neuron (neuron 1) and some target spike trains $Y_l^K$ $(l > 1)$ has to be maximized, while simultaneously minimizing the mutual information between the inputs $X^K$ and the output $Y_1^K$. Obviously, this is the generic IB scenario applied to spiking neurons (see Figure 1A). A learning rule for extracting independent components with spiking neurons (see section 5) can be derived in a similar manner.

For simplicity, we consider the case of an IB optimization for only one target spike train $Y_2^K$, and derive an update rule for the synaptic weights $w_{1j}$ of neuron 1. The quantity to maximize is therefore

$$L = -I(\mathbf{X}^K; \mathbf{Y}_1^K) + \beta I(\mathbf{Y}_1^K; \mathbf{Y}_2^K) - \gamma D_{KL}(P(Y_1^K) || \tilde{P}(Y_1^K)), \tag{7}$$

where $\beta$ and $\gamma$ are optimization constants. To maximize this objective function, we derive the weight change $\Delta w_{1j}^k$ during the $k$-th time step by gradient ascent on (7), assuming that the weights $w_{1j}$ can change between some bounds $0 \leq w_{1j} \leq w_{max}$ (we assume $w_{max} = 1$ throughout this paper).

Note that all three terms of (7) implicitly depend on $w_{1j}$ because the output distribution $P(Y_1^K)$ changes if we modify the weights $w_{1j}$. Since the first and the last term of (7) have already been considered (up to the sign) in [3], we will concentrate here on the middle term $L_{12} := \beta I(\mathbf{Y}_1^K; \mathbf{Y}_2^K)$ and denote the contribution of the gradient of $L_{12}$ to the total weight change $\Delta w_{1j}^k$ in the $k$-th time step by $\Delta \tilde{w}_{1j}^k$.

In order to get an expression for the weight change in a specific time step $t^k$ we write the probabilities $P(Y_i^K)$ and $P(Y_1^K, Y_2^K)$ occurring in (7) as products over individual time bins, i.e., $P(Y_i^K) = \prod_{k=1}^K P(y_i^k | Y_i^{k-1})$ and $P(Y_1^K, Y_2^K) = \prod_{k=1}^K P(y_1^k, y_2^k | Y_1^{k-1}, Y_2^{k-1})$, according to the chain rule of information theory [5]. Consequently, we rewrite $L_{12}$ as a sum over the contributions of the individual time bins, $L_{12} = \sum_{k=1}^K \Delta L_{12}^k$, with

$$\Delta L_{12}^k = \left\langle \beta \log \frac{P(y_1^k, y_2^k | Y_1^{k-1}, Y_2^{k-1})}{P(y_1^k | Y_1^{k-1}) P(y_2^k | Y_2^{k-1})} \right\rangle_{\mathbf{X}^k, \mathbf{Y}_1^k, \mathbf{Y}_2^k}. \tag{8}$$

The weight change $\Delta \tilde{w}_{1j}^k$ is then proportional to the gradient of this expression with respect to the weights $w_{1j}$, i.e., $\Delta \tilde{w}_{1j}^k = \alpha(\partial \Delta L_{12}^k / \partial w_{1j})$, with some learning rate $\alpha > 0$. The evaluation of the gradient yields $\Delta \tilde{w}_{1j}^k = \alpha \left\langle C_{1j}^k \beta F_{12}^k \right\rangle_{\mathbf{X}^k, \mathbf{Y}_1^k, \mathbf{Y}_2^k}$ with a correlation term $C_{1j}^k$ as in (5) and a term

$$F_{12}^k = y_1^k y_2^k \log \frac{\bar{g}_{12}(t^k)}{\bar{g}_1(t^k) \bar{g}_2(t^k)} - y_1^k (1 - y_2^k) R_2(t^k) \Delta t \left[ \frac{\bar{g}_{12}(t^k)}{\bar{g}_1(t^k)} - \bar{g}_2(t^k) \right] -$$

$$- (1 - y_1^k) y_2^k R_1(t^k) \Delta t \left[ \frac{\bar{g}_{12}(t^k)}{\bar{g}_2(t^k)} - \bar{g}_1(t^k) \right] +$$

$$+ (1 - y_1^k)(1 - y_2^k) R_1(t^k) R_2(t^k) (\Delta t)^2 \left[ \bar{g}_{12}(t^k) - \bar{g}_1(t^k) \bar{g}_2(t^k) \right]. \tag{9}$$

Here, $\bar{g}_i(t^k) = \langle g(u_i(t^k)) \rangle_{\mathbf{X}^k | Y_i^{k-1}}$ denotes the average firing rate of neuron $i$ and $\bar{g}_{12}(t^k) = \langle g(u_1(t^k)) g(u_2(t^k)) \rangle_{\mathbf{X}^k | Y_1^{k-1}, Y_2^{k-1}}$ denotes the average product of firing rates of both neurons. Both quantities are implemented online as running exponential averages with a time constant of 10s.

Under the assumption of a small learning rate $\alpha$ we can approximate the expectation $\langle \cdot \rangle_{\mathbf{X}^k, \mathbf{Y}_1^k, \mathbf{Y}_2^k}$ by averaging over a single long trial. Considering now all three terms in (7) we finally arrive at an online rule for maximizing (7)

$$\frac{\Delta w_{1j}^k}{\Delta t} = -\alpha C_{1j}^k \left[ B_1^k(-\gamma) - \beta \Delta t B_{12}^k \right]. \tag{10}$$

which consists of a term $C_{1j}^k$ sensitive to correlations between the output of the neuron and its presynaptic input at synapse $j$ ("correlation term") and terms $B_1^k$ and $B_{12}^k$ that characterize the postsynaptic state of the neuron ("postsynaptic terms"). Note that the argument of $B_1^k$ is different from (4) because some of the terms of the objective function (7) have a different sign. In order to compensate the effect of a small $\Delta t$, the constant $\beta$ has to be large enough for the term $B_{12}^k$ to have an influence on the weight change.

The factors $C_{1j}^k$ and $B_1^k$ were described in the previous section. In addition, our learning rule contains an extra term $B_{12}^k = F_{12}^k / (\Delta t)^2$ that is sensitive to the statistical dependence between the output spike train of the neuron and the target. It is given by

$$B_{12}^k = \frac{y_1^k y_2^k}{(\Delta t)^2} \log \frac{\bar{g}_{12}(t^k)}{\bar{g}_1(t^k) \bar{g}_2(t^k)} - \frac{y_1^k}{\Delta t}(1 - y_2^k) R_2(t^k) \left[ \frac{\bar{g}_{12}(t^k)}{\bar{g}_1(t^k)} - \bar{g}_2(t^k) \right]$$

$$- \frac{y_2^k}{\Delta t}(1 - y_1^k) R_1(t^k) \left[ \frac{\bar{g}_{12}(t^k)}{\bar{g}_2(t^k)} - \bar{g}_1(t^k) \right]$$

$$+ (1 - y_1^k)(1 - y_2^k) R_1(t^k) R_2(t^k) \left[ \bar{g}_{12}(t^k) - \bar{g}_1(t^k) \bar{g}_2(t^k) \right]. \tag{11}$$

This term basically compares the average product of firing rates $\bar{g}_{12}$ (which corresponds to the joint probability of spiking) with the product of average firing rates $\bar{g}_1 \bar{g}_2$ (representing the probability of independent spiking). In this way, it measures the momentary mutual information between the output of the neuron and the target spike train.

For a simplified neuron model without refractoriness ($R(t) = 1$), the update rule (4) resembles the BCM-rule [6] as shown in [3]. With the objective function (7) to maximize, we expect an "anti-Hebbian BCM" rule with another term accounting for statistical dependencies between $Y_1^K$ and $Y_2^K$. Since there is no refractoriness, the postsynaptic rate $\nu_1(t^k)$ is given directly by the current value of $g(u(t^k))$, and the update rule (10) reduces to the rate model[3]

$$\frac{\Delta w_{1j}^k}{\Delta t} = -\alpha \nu_j^{pre,k} f(\nu_1^k) \left\{ \log \left[ \frac{\nu_1^k}{\bar{\nu}_1^k} \left( \frac{\bar{\nu}_1^k}{\tilde{g}} \right)^\gamma \right] \right.$$
$$\left. - \beta \Delta t \left( \nu_2^k \log \left[ \frac{\bar{\nu}_{12}^k}{\bar{\nu}_1^k \bar{\nu}_2^k} \right] - \bar{\nu}_2^k \left[ \frac{\bar{\nu}_{12}^k}{\bar{\nu}_1^k \bar{\nu}_2^k} - 1 \right] \right) \right\}, \quad (12)$$

where the presynaptic rate at synapse $j$ at time $t^k$ is denoted by $\nu_j^{pre,k} = a \sum_{n=1}^k \epsilon(t^k - t^n) x_j^n$ with $a$ in units $(Vs)^{-1}$. The values $\bar{\nu}_1^k$, $\bar{\nu}_2^k$, and $\bar{\nu}_{12}^k$ are running averages of the output rate $\nu_1^k$, the rate of the target signal $\nu_2^k$ and of the product of these values, $\nu_1^k \nu_2^k$, respectively. The function $f(\nu_1^k) = g'(g^{-1}(\nu_1^k))/a$ is proportional to the derivative of $g$ with respect to $u$, evaluated at the current membrane potential. The first term in the curly brackets accounts for the homeostatic process (similar to the BCM rule, see [3]), whereas the second term reinforces dependencies between $Y_1^K$ and $Y_2^K$. Note that this term is zero if the rates of the two neurons are independent.

It is interesting to note that if we rewrite the simplified rate-based learning rule (12) in the following way,

$$\frac{\Delta w_{1j}^k}{\Delta t} = -\alpha \nu_j^{pre,k} \Phi(\nu_1^k, \nu_2^k), \quad (13)$$

we can view it as an extension of the classical Bienenstock-Cooper-Munro (BCM) rule [6] with a two-dimensional synaptic modification function $\Phi(\nu_1^k, \nu_2^k)$. Here, values of $\Phi > 0$ produce LTD whereas values of $\Phi < 0$ produce LTP. These regimes are separated by a sliding threshold, however, in contrast to the original BCM rule this threshold does not only depend on the running average of the postsynaptic rate $\bar{\nu}_1^k$, but also on the current values of $\nu_2^k$ and $\bar{\nu}_2^k$.

## 4  Application to Information Bottleneck Optimization

We use a setup as in Figure 1A where we want to maximize the information which the output $Y_1^K$ of a learning neuron conveys about two target signals $Y_2^K$ and $Y_3^K$. If the target signals are statistically independent from each other we can optimize the mutual information to each target signal separately. This leads to an update rule

$$\frac{\Delta w_{1j}^k}{\Delta t} = -\alpha C_{1j}^k \left[ B_1^k(-\gamma) - \beta \Delta t \left( B_{12}^k + B_{13}^k \right) \right], \quad (14)$$

where $B_{12}^k$ and $B_{13}^k$ are the postsynaptic terms (11) sensitive to the statistical dependence between the output and target signals 1 and 2, respectively. We choose $\tilde{g} = 30$Hz for the target firing rate, and we use discrete time with $\Delta t = 1$ms.

In this experiment we demonstrate that it is possible to consider two very different kinds of target signals: one target spike train has has a similar rate modulation as one part of the input, while the other target spike train has a high spike-spike correlation with another part of the input. The learning neuron receives input at 100 synapses, which are divided into 4 groups of 25 inputs each. The first two input groups consist of rate modulated Poisson spike trains[4] (Figure 2A). Spike trains from the remaining groups 3 and 4 are correlated with a coefficient of 0.5 within each group, however, spike trains from different groups are uncorrelated. Correlated spike trains are generated by the procedure described in [7].

The first target signal is chosen to have the same rate modulation as the inputs from group 1, except that Gaussian random noise is superimposed with a standard deviation of 2Hz. The second target spike train is correlated with inputs from group 3 (with a coefficient of 0.5), but uncorrelated to inputs from group 4. Furthermore, both target signals are silent during random intervals: at each

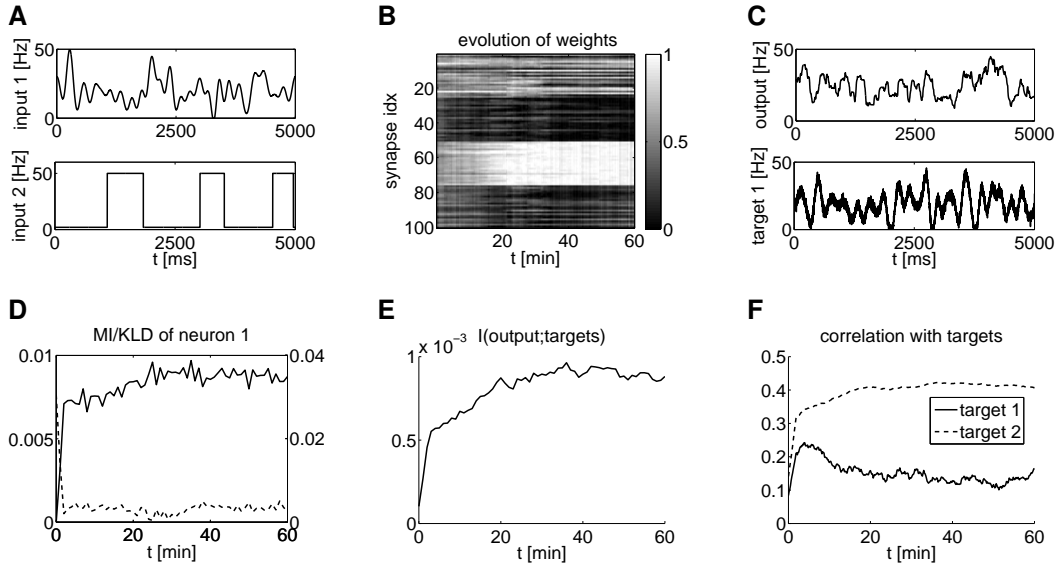

Figure 2: Performance of the spike-based learning rule (10) for the IB task. **A** Modulation of input rates to input groups 1 and 2. **B** Evolution of weights during 60 minutes of learning (bright: strong synapses, $w_{ij} \approx 1$, dark: depressed synapses, $w_{ij} \approx 0$.) Weights are initialized randomly between 0.10 and 0.12, $\alpha = 10^{-4}$, $\beta = 2 \cdot 10^3$, $\gamma = 50$. **C** Output rate and rate of target signal 1 during 5 seconds after learning. **D** Evolution of the average mutual information per time bin (solid line, left scale) between input and output and the Kullback-Leibler divergence per time bin (dashed line, right scale) as a function of time. Averages are calculated over segments of 1 minute. **E** Evolution of the average mutual information per time bin between output and both target spike trains as a function of time. **F** Trace of the correlation between output rate and rate of target signal 1 (solid line) and the spike-spike correlation (dashed line) between the output and target spike train 2 during learning. Correlation coefficients are calculated every 10 seconds.

time step, each target signal is independently set to 0 with a certain probability ($10^{-5}$) and remains silent for a duration chosen from a Gaussian distribution with mean 5s and SD 1s (minimum duration is 1s). Hence this experiment tests whether learning works even if the target signals are not available all of the time.

Figure 2 shows that strong weights evolve for the first and third group of synapses, whereas the efficacies for the remaining inputs are depressed. Both groups with growing weights are correlated with one of the target signals, therefore the mutual information between output and target spike trains increases. Since spike-spike correlations convey more information than rate modulations synaptic efficacies develop more strongly to group 3 (the group with spike-spike correlations). This results in an initial decrease in correlation with the rate-modulated target to the benefit of higher correlation with the second target. However, after about 30 minutes when the weights become stable, the correlations as well as the mutual information quantities stay roughly constant.

An application of the simplified rule (12) to the same task is shown in Figure 3 where it can be seen that strong weights close to $w_{max}$ are developed for the rate-modulated input. To some extent weights grow also for the inputs with spike-spike correlations in order to reach the constant target firing rate $\tilde{g}$. In contrast to the spike-based rule the simplified rule is not able to detect spike-spike correlations between output and target spike trains.

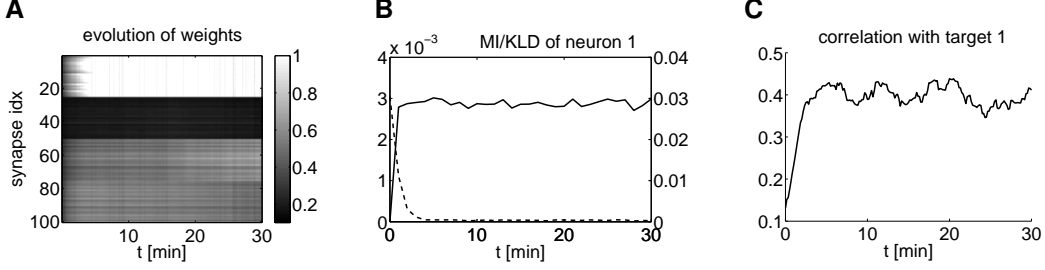

Figure 3: Performance of the simplified update rule (12) for the IB task. **A** Evolution of weights during 30 minutes of learning (bright: strong synapses, $w_{ij} \approx 1$, dark: depressed synapses, $w_{ij} \approx 0$.) Weights are initialized randomly between 0.10 and 0.12, $\alpha = 10^{-3}$, $\beta = 10^4$, $\gamma = 10$. **B** Evolution of the average mutual information per time bin (solid line, left scale) between input and output and the Kullback-Leibler divergence per time bin (dashed line, right scale) as a function of time. Averages are calculated over segments of 1 minute. **C** Trace of the correlation between output rate and target rate during learning. Correlation coefficients are calculated every 10 seconds.

## 5  Extracting Independent Components

With a slight modification in the objective function (7) the learning rule allows us to extract statistically independent components from an ensemble of input spike trains. We consider two neurons receiving the same input at their synapses (see Figure 1B). For both neurons $i = 1, 2$ we maximize information transmission under the constraint that their outputs stay as statistically independent from each other as possible. That is, we maximize

$$\tilde{L}_i = I(\mathbf{X}^K; \mathbf{Y}_i^K) - \beta I(\mathbf{Y}_1^K; \mathbf{Y}_2^K) - \gamma D_{KL}(P(Y_i^K)\|\tilde{P}(Y_i^K)). \tag{15}$$

Since the same terms (up to the sign) are optimized in (7) and (15) we can derive a gradient ascent rule for the weights of neuron $i$, $w_{ij}$, analogously to section 3:

$$\frac{\Delta w_{ij}^k}{\Delta t} = \alpha C_{ij}^k \left[ B_i^k(\gamma) - \beta \Delta t B_{12}^k \right]. \tag{16}$$

Figure 4 shows the results of an experiment where two neurons receive the same Poisson input with a rate of 20Hz at their 100 synapses. The input is divided into two groups of 40 spike trains each, such that synapses 1 to 40 and 41 to 80 receive correlated input with a correlation coefficient of 0.5 within each group, however, any spike trains belonging to different input groups are uncorrelated. The remaining 20 synapses receive uncorrelated Poisson input. Weights close to the maximal efficacy $w_{max} = 1$ are developed for one of the groups of synapses that receives correlated input (group 2 in this case) whereas those for the other correlated group (group 1) as well as those for the uncorrelated group (group 3) stay low. Neuron 2 develops strong weights to the other correlated group of synapses (group 1) whereas the efficacies of the second correlated group (group 2) remain depressed, thereby trying to produce a statistically independent output. For both neurons the mutual information is maximized and the target output distribution of a constant firing rate of 30Hz is approached well. After an initial increase in the mutual information and in the correlation between the outputs, when the weights of both neurons start to grow simultaneously, the amounts of information and correlation drop as both neurons develop strong efficacies to different parts of the input.

## 6  Discussion

Information Bottleneck (IB) and Independent Component Analysis (ICA) have been proposed as general principles for unsupervised learning in lower cortical areas, however, learning rules that can implement these principles with spiking neurons have been missing. In this article we have derived from information theoretic principles learning rules which enable a stochastically spiking neuron to solve these tasks. These learning rules are optimal from the perspective of information theory, but they are not local in the sense that they use only information that is available at a single

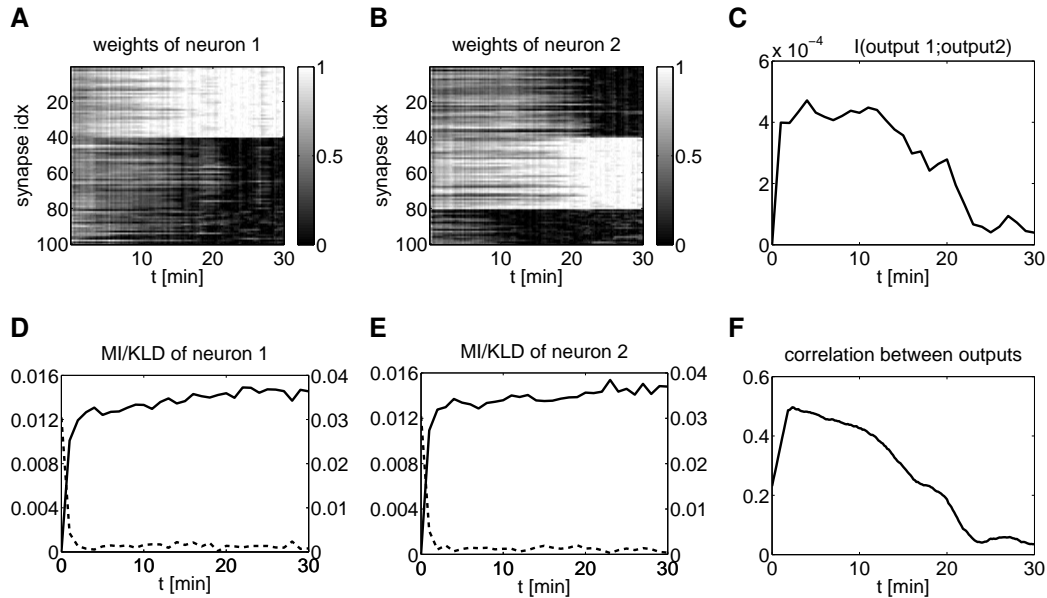

Figure 4: Extracting independent components. **A,B** Evolution of weights during 30 minutes of learning for both postsynaptic neurons (red: strong synapses, $w_{ij} \approx 1$, blue: depressed synapses, $w_{ij} \approx 0$.) Weights are initialized randomly between 0.10 and 0.12, $\alpha = 10^{-3}$, $\beta = 100$, $\gamma = 10$. **C** Evolution of the average mutual information per time bin between both output spike trains as a function of time. **D,E** Evolution of the average mutual information per time bin (solid line, left scale) between input and output and the Kullback-Leibler divergence per time bin for both neurons (dashed line, right scale) as a function of time. Averages are calculated over segments of 1 minute. **F** Trace of the correlation between both output spike trains during learning. Correlation coefficients are calculated every 10 seconds.

synapse without an auxiliary network of interneurons or other biological processes. Rather, they tell us what type of information would have to be ideally provided by such auxiliary network, and how the synapse should change its efficacy in order to approximate a theoretically optimal learning rule.

## Acknowledgments

We would like to thank Wulfram Gerstner and Jean-Pascal Pfister for helpful discussions. This paper was written under partial support by the Austrian Science Fund FWF, # S9102-N13 and # P17229-N04, and was also supported by PASCAL, project # IST2002-506778, and FACETS, project # 15879, of the European Union.

## Footnotes

[1] We use boldface letters ($\mathbf{X}^k$) to distinguish random variables from specific realizations ($X^k$).

[2] The rate $\bar{g}_i(t^k) = \langle g(u_i(t^k)) \rangle_{\mathbf{X}^k | Y_i^{k-1}}$ denotes an expectation of the firing rate over the input distribution given the postsynaptic history and is implemented as a running average with an exponential time window (with a time constant of 10ms).

[3]In the absence of refractoriness we use an alternative gain function $g_{alt}(u) = [1/g_{max} + 1/g(u)]^{-1}$ in order to pose an upper limit of $g_{max} = 100$Hz on the postsynaptic firing rate.

[4]The rate of the first 25 inputs is modulated by a Gaussian white-noise signal with mean 20Hz that has been low pass filtered with a cut-off frequency of 5Hz. Synapses 26 to 50 receive a rate that has a constant value of 2Hz, except that a burst is initiated at each time step with a probability of 0.0005. Thus there is a burst on average every 2s. The duration of a burst is chosen from a Gaussian distribution with mean 0.5s and SD 0.2s, the minimum duration is chosen to be 0.1s. During a burst the rate is set to 50Hz. In the simulations we use discrete time with $\Delta t = 1$ms.

## References

[1] N. Tishby, F. C. Pereira, and W. Bialek. The information bottleneck method. In *Proceedings of the 37-th Annual Allerton Conference on Communication, Control and Computing*, pages 368–377, 1999.

[2] A. Hyvärinen, J. Karhunen, and E. Oja. *Independent Component Analysis*. Wiley, New York, 2001.

[3] T. Toyoizumi, J.-P. Pfister, K. Aihara, and W. Gerstner. Generalized Bienenstock-Cooper-Munro rule for spiking neurons that maximizes information transmission. *Proc. Natl. Acad. Sci. USA*, 102:5239–5244, 2005.

[4] W. Gerstner and W. M. Kistler. *Spiking Neuron Models*. Cambridge University Press, Cambridge, 2002.

[5] T. M. Cover and J. A. Thomas. *Elements of Information Theory*. Wiley, New York, 1991.

[6] E. L. Bienenstock, L. N. Cooper, and P. W. Munro. Theory for the development of neuron selectivity: orientation specificity and binocular interaction in visual cortex. *J. Neurosci.*, 2(1):32–48, 1982.

[7] R. Gütig, R. Aharonov, S. Rotter, and H. Sompolinsky. Learning input correlations through non-linear temporally asymmetric hebbian plasticity. *Journal of Neurosci.*, 23:3697–3714, 2003.
